# Beyond Spectral Clustering - Tight Relaxations of Balanced Graph Cuts

**Matthias Hein**
Saarland University, Saarbrücken, Germany
hein@cs.uni-saarland.de

**Simon Setzer**
Saarland University, Saarbrücken, Germany
setzer@mia.uni-saarland.de

## Abstract

Spectral clustering is based on the spectral relaxation of the normalized/ratio graph cut criterion. While the spectral relaxation is known to be loose, it has been shown recently that a non-linear eigenproblem yields a tight relaxation of the Cheeger cut. In this paper, we extend this result considerably by providing a characterization of all balanced graph cuts which allow for a tight relaxation. Although the resulting optimization problems are non-convex and non-smooth, we provide an efficient first-order scheme which scales to large graphs. Moreover, our approach comes with the quality guarantee that given any partition as initialization the algorithm either outputs a better partition or it stops immediately.

## 1 Introduction

The problem of finding the best balanced cut of a graph is an important problem in computer science [9, 24, 13]. It has been used for minimizing the communication cost in parallel computing, reordering of sparse matrices, image segmentation and clustering. In particular, in machine learning spectral clustering is one of the most popular graph-based clustering methods as it can be applied to any graph-based data or to data where similarity information is available so that one can build a neighborhood graph. Spectral clustering is originally based on a relaxation of the combinatorial normalized/ratio graph cut problem, see [28]. The relaxation with the best known worst case approximation guarantee yields a semi-definite program, see [3]. However, it is practically infeasible for graphs with more than 100 vertices due to the presence of $O(n^3)$ constraints where $n$ is the number of vertices in the graph. In contrast, the computation of eigenvectors of a sparse graph scales easily to large graphs. In a line of recent work [6, 26, 14] it has been shown that relaxation based on the nonlinear graph $p$-Laplacian lead to similar runtime performance while providing much better cuts. In particular, for $p = 1$ one obtains a tight relaxation of the Cheeger cut, see [8, 26, 14].

In this work, we generalize this result considerably. Namely, we provide for almost any balanced graph cut problem a tight relaxation into a continuous problem. This allows flexible modeling of different graph cut criteria. The resulting non-convex, non-smooth continuous optimization problem can be efficiently solved by our new method for the minimization of ratios of differences of convex functions, called RatioDCA. Moreover, compared to [14], we also provide a more efficient way how to solve the resulting convex inner problems by transferring recent methods from total variation denoising, cf. [7], to the graph setting. In first experiments, we illustrate the effect of different balancing terms and show improved clustering results of USPS and MNIST compared to [14].

## 2 Set Functions, Submodularity, Convexity and the Lovasz Extension

In this section we gather some material from the literature on set functions, submodularity and the Lovasz extension, which we need in the next section. We refer the reader to [11, 4] for a more detailed exposition. We work on weighted, undirected graphs $G = (V, W)$ with vertex set $V$ and

a symmetric, non-negative weight matrix $W$. We define $n := |V|$ and denote by $\overline{A} = V \backslash A$ the complement of $A$ in $V$, set functions are denoted with a hat, $\hat{S}$, whereas the corresponding Lovasz extension is simply $S$. The indicator vector of a set $A$ is written as $\mathbf{1}_A$. In the following we always assume that for any considered set function $\hat{S}$ it holds $\hat{S}(\emptyset) = 0$. The Lovasz extension is a way to extend a set function from $2^V$ to $\mathbb{R}^V$.

**Definition 2.1** *Let* $\hat{S} : 2^V \to \mathbb{R}$ *be a set function with* $\hat{S}(\emptyset) = 0$. *Let* $f \in \mathbb{R}^V$ *be ordered in increasing order* $f_1 \le f_2 \le \ldots \le f_n$ *and define* $C_i = \{j \in V \mid f_j > f_i\}$ *where* $C_0 = V$. *Then* $S : \mathbb{R}^V \to \mathbb{R}$ *given by*

$$S(f) = \sum_{i=1}^n f_i \Big( \hat{S}(C_{i-1}) - \hat{S}(C_i) \Big) = \sum_{i=1}^{n-1} \hat{S}(C_i)(f_{i+1} - f_i) + f_1 \hat{S}(V)$$

*is called the **Lovasz extension** of* $\hat{S}$. *Note that* $S(\mathbf{1}_A) = \hat{S}(A)$ *for all* $A \subset V$.

Note that for symmetric set functions $\hat{S}$, that is $\hat{S}(A) = \hat{S}(\overline{A})$ for all $A \subset V$, the property $\hat{S}(\emptyset) = 0$ implies $\hat{S}(V) = 0$. A particular interesting class of set functions are the submodular set functions as their Lovasz extension is convex.

**Definition 2.2** *A set function,* $\hat{F} : 2^V \to \mathbb{R}$ *is **submodular** if for all* $A, B \subset V$,

$$\hat{F}(A \cup B) + \hat{F}(A \cap B) \le \hat{F}(A) + \hat{F}(B).$$

$\hat{F}$ *is called **strictly submodular** if the inequality is strict whenever* $A \not\subseteq B$ *or* $B \not\subseteq A$.

Note that symmetric submodular set functions are always non-negative as for all $A \subset V$,

$$2\hat{F}(A) = \hat{F}(A) + \hat{F}(\overline{A}) \ge \hat{F}(A \cup \overline{A}) + \hat{F}(A \cap \overline{A}) = \hat{F}(V) + \hat{F}(\emptyset) = 0.$$

An important class of set functions for clustering are cardinality-based set functions.

**Proposition 2.1 ([4])** *Let* $e \in \mathbb{R}_+^V$ *and* $g : \mathbb{R}_+ \to \mathbb{R}$ *is a concave function, then* $\hat{F} : A \mapsto g(s(A))$ *is submodular. If* $\hat{F} : A \mapsto g(s(A))$ *is submodular for all* $s \in \mathbb{R}_+^V$, *then* $g$ *is concave.*

The following properties hold for the Lovasz extension.

**Proposition 2.2 ([11, 4])** *Let* $S : \mathbb{R}^V \to \mathbb{R}$ *be the Lovasz extension of* $\hat{S} : 2^V \to \mathbb{R}$ *with* $\hat{S}(\emptyset) = 0$.

- $\hat{S}$ *is submodular if and only if* $S$ *is convex,*
- $S$ *is positively one-homogeneous,*
- $S(f) \ge 0, \ \forall f \in \mathbb{R}^V$ *and* $S(\mathbf{1}) = 0$ *if and only if* $\hat{S}(A) \ge 0, \ \forall A \subset V$ *and* $\hat{S}(V) = 0$.
- $S(f + \alpha \mathbf{1}) = S(f)$ *for all* $f \in \mathbb{R}^V, \alpha \in \mathbb{R}$ *if and only if* $\hat{S}(V) = 0$,
- $S$ *is even, if* $\hat{S}$ *is symmetric.*

One might wonder if the Lovasz extension of all submodular set functions generates the set of all positively one-homogeneous convex functions. This is not the case, as already Lovasz [19] gave a counter-example. In the next section we will be interested in the class of positively one-homogeneous, even, convex functions $S$ with $S(f + \alpha \mathbf{1}) = S(f)$ for all $f \in \mathbb{R}^V$. From the above proposition we deduce that these properties are fulfilled for the Lovasz extension of any symmetric, submodular set function. However, also for this special class there exists a counter-example. Take

$$S(f) = \left\| f - \frac{1}{|V|} \langle f, \mathbf{1} \rangle \mathbf{1} \right\|_\infty.$$

It fulfills all the stated conditions but it induces the set function $\hat{S}(A) := S(\mathbf{1}_A)$ given as

$$\hat{S}(A) = \frac{1}{|V|} \begin{cases} \max\{|A|, |V \backslash A|\}, & 0 < |A| < |V| \\ 0, & \text{else} \end{cases}$$

It is easy to check that this function is not submodular. Thus different convex one-homogeneous functions can induce the same set function via $\hat{S}(A) := S(\mathbf{1}_A)$.

It is known [15] that a large class of functions e.g. every $f \in C^2(\mathbb{R}^n)$ can be written as a difference of convex functions. As submodular functions correspond to convex functions in the sense of the Lovasz extension, one can ask if the same result holds for set functions: Is every set function a difference of submodular set functions ? The following result has been reported in [21]. As some properties assumed in the proof in [21] do not hold, we give an alternative constructive proof.

**Proposition 2.3** *Every set function $\hat{S} : 2^V \to \mathbb{R}$ can be written as the difference of two submodular functions. The corresponding Lovasz extension $S : \mathbb{R}^V \to \mathbb{R}$ can be written as a difference of convex functions.*

Note that the proof of Proposition 2.3 is constructive. Thus we can always find the decomposition of the set function into a difference of two submodular functions and thus also the decomposition of its Lovasz extension into a difference of convex functions.

## 3 Tight Relaxations of Balanced Graph Cuts

In graph-based clustering a popular criterion to partition the graph is to minimize the cut $\mathrm{cut}(A, \overline{A})$, defined as

$$\mathrm{cut}(A, \overline{A}) = \sum_{i \in A, j \in \overline{A}} w_{ij},$$

where $(w_{ij}) \in \mathbb{R}^{|V| \times |V|}$ are the non-negative, symmetric weights of the undirected graph $G = (V, W)$ usually interpreted as similarities of vertices $i$ and $j$. Direct minimization of the cut leads typically to very unbalanced partitions, where often just a single vertex is split off. Therefore one has to introduce a balancing term which biases the criterion towards balanced partitions. Two popular balanced graph cut criterion are the Cheeger cut $\mathrm{RCC}(A, \overline{A})$ and the ratio cut $\mathrm{RCut}(A, \overline{A})$

$$\mathrm{RCC}(A, \overline{A}) = \frac{\mathrm{cut}(A, \overline{A})}{\min\{|A|, |\overline{A}|\}}, \qquad \mathrm{RCut}(A, \overline{A}) = |V| \frac{\mathrm{cut}(A, \overline{A})}{|A||\overline{A}|} = \mathrm{cut}(A, \overline{A}) \Big( \frac{1}{|A|} + \frac{1}{|\overline{A}|} \Big).$$

We consider later on also their normalized versions. Spectral clustering is derived as relaxation of the ratio cut criterion based on the second eigenvector of the graph Laplacian. While the second eigenvector can be efficiently computed, it is well-known that this relaxation is far from being tight. In particular there exist graphs where the spectral relaxation is as bad [12] as the isoperimetric inequality suggests [1]. In a recent line of work [6, 26, 14] it has been shown that a tight relaxation for the Cheeger cut can be achieved by moving from the linear eigenproblem to a nonlinear eigenproblem associated to the nonlinear graph 1-Laplacian [14].

In this work we generalize this result considerably by showing in Theorem 3.1 that a tight relaxation exists for every balanced graph cut measure which is of the form cut divided by balancing term. More precisely, let $\hat{S} : 2^V \to \mathbb{R}$ be a symmetric non-negative set function. Then a **balanced graph cut criterion** $\phi : 2^V \to \mathbb{R}_+$ of a partition $(A, \overline{A})$ has the form,

$$\phi(A) := \frac{\mathrm{cut}(A, \overline{A})}{\hat{S}(A)}. \tag{1}$$

As we consider undirected graphs, the cut is a symmetric set function and thus $\phi(A) = \phi(\overline{A})$. In order to get a balanced graph cut, $\hat{S}$ is typically chosen as a function of $|A|$ (or some other type of volume) which is monotonically increasing on $[0, |V|/2]$. The first part of the theorem showing the equivalence of combinatorial and continuous problem is motivated by a result derived by Rothaus in [25] in the context of isoperimetric inequalities on Riemannian manifolds. It has been transferred to graphs by Tillich and independently by Houdre in [27, 16]. We generalize their result further so that it now holds for all possible non-negative symmetric set functions. In order to establish the link to the result of Rothaus, we first state the following characterization

**Lemma 3.1** *A function $S : V \to \mathbb{R}$ is positively one-homogeneous, even, convex and $S(f + \alpha \mathbf{1}) = S(f)$ for all $f \in \mathbb{R}^V, \alpha \in \mathbb{R}$ if and only if $S(f) = \sup_{u \in U} \langle u, f \rangle$ where $U \subset \mathbb{R}^n$ is a closed symmetric convex set and $\langle u, \mathbf{1} \rangle = 0$ for any $u \in U$.*

**Theorem 3.1** *Let $G = (V, E)$ be a finite, weighted undirected graph and $S : \mathbb{R}^V \to \mathbb{R}$ and let $\hat{S} : 2^V \to \mathbb{R}$ be symmetric with $\hat{S}(\emptyset) = 0$, then*

$$\inf_{f \in \mathbb{R}^V} \frac{\frac{1}{2} \sum_{i,j=1}^n w_{ij} |f_i - f_j|}{S(f)} = \inf_{A \subset V} \frac{\mathrm{cut}(A, \overline{A})}{\hat{S}(A)},$$

*if either one of the following two conditions holds*

1. *$S$ is positively one-homogeneous, even, convex and $S(f + \alpha \mathbf{1}) = S(f)$ for all $f \in \mathbb{R}^V$, $\alpha \in \mathbb{R}$ and $\hat{S}$ is defined as $\hat{S}(A) := S(\mathbf{1}_A)$ for all $A \subset V$.*

2. *$S$ is the Lovasz extension of the non-negative, symmetric set function $\hat{S}$ with $\hat{S}(\emptyset) = 0$.*

*Let $f \in \mathbb{R}^V$ and denote by $C_t := \{i \in V \mid f_i > t\}$, then it holds under both conditions,*

$$\min_{t \in \mathbb{R}} \frac{\mathrm{cut}(C_t, \overline{C_t})}{\hat{S}(C_t)} \leq \frac{\frac{1}{2} \sum_{i,j=1}^n w_{ij} |f_i - f_j|}{S(f)}.$$

Theorem 3.1 can be generalized by replacing the cut with an arbitrary other set function. However, the emphasis of this paper is to use the new degree of freedom for balanced graph clustering. The more general approach will be discussed elsewhere. Note that the first condition in Theorem 3.1 implies that $\hat{S}$ is symmetric as

$$\hat{S}(A) = S(\mathbf{1}_A) = S(-\mathbf{1}_A) = S(\mathbf{1} - \mathbf{1}_A) = S(\mathbf{1}_{\overline{A}}) = \hat{S}(\overline{A}).$$

Moreover, $\hat{S}$ is non-negative with $\hat{S}(\emptyset) = \hat{S}(V) = 0$ as $S$ is even, convex and positively one-homogeneous. For the second condition note that by Proposition 2.3 the Lovasz extension of any set function can be written as a difference of convex (d.c.) functions. As the total variation term in the enumerator is convex, we thus have to minimize a ratio of a convex and a d.c. function. The efficient minimization of such problems will be the topic of the next section.

We would like to point out a related line of work for the case where the balancing term $\hat{S}$ is submodular and the balanced graph cut measure is directly optimized using submodular minimization techniques. In [23] this idea is proposed for the ratio cut and subsequently generalized [22, 17] so that every submodular balancing function $\hat{S}$ can be used. While the general framework is appealing, it is unclear if the minimization can be done efficiently. Moreover, note that Theorem 3.1 goes well beyond the case where $\hat{S}$ is submodular.

## 3.1   Examples of Balancing Set Functions

Theorem 3.1 opens up new modeling possibilities for clustering based on balanced graph cuts. We discuss in the experiments differences and properties of the individual balancing terms. However, it is out of the scope of this paper to answer the question which balancing term is the "best". An answer to such a question is likely to be application-dependent. However, for a given random graph model it might be possible to suggest a suitable balancing term given one knows how cut and volume behave. A first step in this direction has been done in [20] where the limit of cut and volume has been discussed for different neighborhood graph types.

In the following we assume that we work with graphs which have non-negative edge weights $W = (w_{ij})$ and non-negative vertex weights $e : V \to \mathbb{R}_+$. The volume $\mathrm{vol}(A)$ of a set $A \subset V$ is defined as $\mathrm{vol}(A) = \sum_{i \in A} e_i$. The volume reduces to the cardinality if $e_i = 1$ for all $i \in V$ (unnormalized case) or to the volume considered in the normalized cut, $\mathrm{vol}(A) = \sum_{i \in A} d_i$ for $e_i = d_i$ for all $i \in V$ (normalized case), where $d_i$ is the degree of vertex $i$. We denote by $E$ the diagonal matrix with $E_{ii} = e_i, i = i, \ldots, n$. Using general vertex weights allows us to present the unnormalized and normalized case in a unified framework. Moreover, general vertex weights allow more modeling freedom e.g. one can give two different vertices very large vertex weights and so implicitly enforce that they will be in different partitions.

| Name | $S(f)$ | $\hat{S}(A)$ |
|---|---|---|
| Cheeger $p$-cut | $\left( \sum\limits_{i=1}^{n} e_i \lvert f_i - \mathrm{wmean}_p(f)\rvert^p \right)^{\frac{1}{p}}$ | $\dfrac{\left(\mathrm{vol}(A)\,\mathrm{vol}(\overline{A})\right)^{\frac{1}{p}}}{\left(\mathrm{vol}(A)^{\frac{1}{p-1}}+\mathrm{vol}(\overline{A})^{\frac{1}{p-1}}\right)^{1-\frac{1}{p}}}$ |
| Normalized $p$-cut | $\left( \sum\limits_{i=1}^{n} e_i \lvert f_i - \frac{\langle e,f\rangle}{\mathrm{vol}(V)}\rvert^p \right)^{\frac{1}{p}}$ | $\dfrac{\left(\mathrm{vol}(A)\,\mathrm{vol}(\overline{A})^p+\mathrm{vol}(A)^p\,\mathrm{vol}(\overline{A})\right)^{\frac{1}{p}}}{\mathrm{vol}(V)}$ |
| Trunc. Cheeger cut | $g_{\max,\alpha}(f) - g_{\min,\alpha}(f)$ | $\begin{cases} \mathrm{vol}(A), & \text{if } \mathrm{vol}(A) \le \alpha\,\mathrm{vol}(V), \\ \mathrm{vol}(\overline{A}), & \text{if } \mathrm{vol}(\overline{A}) \le \alpha\,\mathrm{vol}(V), \\ \alpha\,\mathrm{vol}(V), & \text{else.} \end{cases}$ |
| Hard balanced cut | $\begin{aligned} & \left(g_{\max,\frac{K}{\lvert V\rvert}}(f) - g_{\min,\frac{K}{\lvert V\rvert}}(f)\right) \\ & -\left(g_{\max,\frac{K-1}{\lvert V\rvert}}(f) - g_{\min,\frac{K-1}{\lvert V\rvert}}(f)\right) \end{aligned}$ | $\begin{cases} 1, & \text{if } \min\{\lvert A\rvert,\lvert\overline{A}\rvert\} \ge K \\ 0, & \text{else.} \end{cases}$ |
| Hard Cheeger cut | $\begin{aligned} & \lVert f - \mathrm{median}(f)\mathbf{1}\rVert_1 \\ & -\left(g_{\max,\frac{K-1}{\lvert V\rvert}}(f) - g_{\min,\frac{K-1}{\lvert V\rvert}}(f)\right) \end{aligned}$ | $\begin{cases} 0, & \text{if } \min\{\lvert A\rvert,\lvert\overline{A}\rvert\} < K, \\ \min\{\lvert A\rvert,\lvert\overline{A}\rvert\} & \\ \quad -(K-1), & \text{else.} \end{cases}$ |

Table 1: Examples of balancing set functions and their continuous counterpart. For the hard balanced and hard Cheeger cut we have unit vertex weights, that is $e_i \equiv 1$.

We report here the Lovasz extension of two important set functions which will be needed in the sequel. For that we define the functions $g_{\max,\alpha}$ and $g_{\min,\alpha}$ as:

$$g_{\max,\alpha}(f) = \max \left\{ \langle \rho, f\rangle \ \big| \ 0 \le \rho_i \le e_i, \ \forall i = 1,\dots,n, \sum_{i=1}^{n} \rho_i = \alpha\,\mathrm{vol}(V)\right\},$$

$$g_{\min,\alpha}(f) = \min \left\{ \langle \rho, f\rangle \ \big| \ 0 \le \rho_i \le e_i, \ \forall i = 1,\dots,n, \sum_{i=1}^{n} \rho_i = \alpha\,\mathrm{vol}(V)\right\}$$

and the weighted $p$-mean $\mathrm{wmean}_p(f)$ is defined as $\mathrm{wmean}_p(f) = \inf_{a\in\mathbb{R}} \sum_{i=1}^{n} e_i \lvert f_i - a\rvert^p$. Note that $g_{\max,\alpha}$ is convex, whereas $g_{\min,\alpha}$ is concave. Both functions can be easily evaluated by sorting the componentwise product $e_i f_i$.

**Proposition 3.1** *Let $\hat{S} : 2^V \to \mathbb{R}$, $\hat{S}(A) := \min\{\mathrm{vol}(A), \mathrm{vol}(\overline{A})\}$. Then the Lovasz extension $S : V \to \mathbb{R}$ is given by $S(f) = \lVert E(f - \mathrm{wmean}_1(f)\mathbf{1})\rVert_1$.*

*Let $e_i = 1, \forall i \in V$ and $\hat{S} : 2^V \to \mathbb{R}$, $\hat{S}(A) := \begin{cases} \min\{\lvert A\rvert,\lvert\overline{A}\rvert\}, & \text{if } \min\{\lvert A\rvert,\lvert\overline{A}\rvert\} \le K, \\ K, & \text{else.} \end{cases}$ Then the Lovasz extension $S : V \to \mathbb{R}$ is given as $S(f) = g_{\max,\frac{K}{\lvert V\rvert}}(f) - g_{\min,\frac{K}{\lvert V\rvert}}(f)$.*

In Table 1 we collect a set of interesting set functions enforcing different levels of balancing. For the Cheeger and Normalized $p$-cut family and the truncated Cheeger cut the functions $S$ are convex and not necessarily the Lovasz extension of the induced set functions $\hat{S}$ (first case in Theorem 3.1). In the case of hard balanced and hard Cheeger cut the set function $\hat{S}$ is not submodular. However, in both cases we know an explicit decomposition of the set function $\hat{S}$ into a difference of submodular functions and thus their Lovasz extension $S$ can be written as a difference of the convex functions. The derivations can be found in the supplementary material.

## 4   Minimization of Ratios of Non-negative Differences of Convex Functions

In [14], the problem of computing the optimal Cheeger cut partition is formulated as a nonlinear eigenproblem. Hein and Bühler show that the second eigenvector of the nonlinear 1-graph Laplacian is equal to the indicator function of the optimal partition. In Theorem 3.1, we have generalized this relation considerably. In this section, we discuss the efficient computation of critical points of the continuous ratios of Theorem 3.1. We propose a general scheme called RatioDCA for minimizing ratios of non-negative differences of convex functions and thus generalizes Algorithm 1 of [14] which could handle only ratios of convex functions. As the optimization problem is non-smooth and non-convex, only convergence to critical points can be guaranteed. However, we will show that for every balanced graph cut criterion our algorithm improves a given partition or it terminates directly. Note that such types of algorithms have been considered for specific graph cut criteria [23, 22, 2].

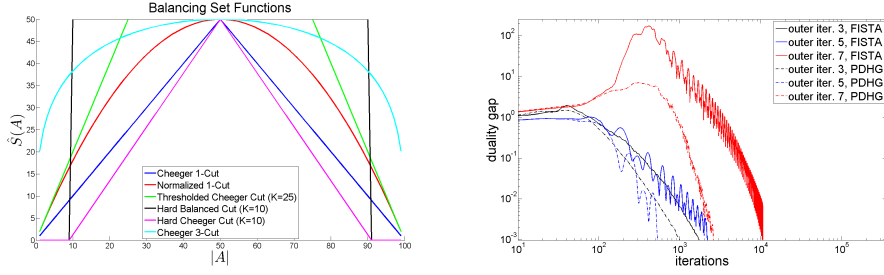

Figure 1: Left: Illustration of different balancing functions (rescaled so that they attain value $|V|/2$ at $|V|/2$). Right: Log-log plot of the duality gap of the inner problem vs. the number of iterations of PDHG (dashed) and FISTA (solid) in outer iterations 3 (black), 5 (blue) and 7 (red) of RatioDCA corresponding to increasing difficulty of the problem. PDHG significantly outperforms FISTA.

## 4.1 General Scheme

The continuous optimization problem in Theorem 3.1 has the form

$$\min_{f \in \mathbb{R}^V} \frac{\frac{1}{2}\sum_{i,j=1}^n w_{ij}|f_i - f_j|}{S(f)}, \tag{2}$$

where $S$ is one-homogeneous and either convex or the Lovasz extension of a non-negative symmetric set function. By Proposition 2.3 the Lovasz extension of any set function can be written as a difference of one-homogeneous convex functions. Using the fourth property of Proposition 2.2 the Lovasz extension $S$ is non-negative, that is $S(f) \geq 0$ for all $f \in \mathbb{R}^V$. With the algorithm RatioDCA below, we provide a general scheme for the minimization of a ratio $F(f) := R(f)/S(f)$, where $R$ and $S$ are non-negative and one-homogeneous and each can be written as a difference of convex functions: $R(f) = R_1(f) - R_2(f)$ and $S(f) = S_1(f) - S_2(f)$ with $R_1, R_2, S_1, S_2$ being convex. In

---

**Algorithm RatioDCA** – Minimization of a non-negative ratio of 1-homogeneous d.c. functions

1: **Initialization:** $f^0 = $ random with $\|f^0\| = 1$, $\lambda^0 = F(f^0)$
2: **repeat**
3:     $s_1(f^k) \in \partial S_1(f^k)$, $r_2(f^k) \in \partial R_2(f^k)$
4:     $f^{k+1} = \underset{\|u\|_2 \leq 1}{\arg\min} \left\{ R_1(u) - \langle u, r_2(f^k)\rangle + \lambda^k \left( S_2(u) - \langle u, s_1(f^k)\rangle \right) \right\}$
5:     $\lambda^{k+1} = (R_1(f^{k+1}) - R_2(f^{k+1}))/(S_1(f^{k+1}) - S_2(f^{k+1}))$
6: **until** $\frac{|\lambda^{k+1} - \lambda^k|}{\lambda^k} < \epsilon$
7: **Output:** eigenvalue $\lambda^{k+1}$ and eigenvector $f^{k+1}$.

---

our setting $R(f) = R_1(f) = \frac{1}{2}\sum_{i,j=1}^n w_{i,j}|f_i - f_j|$. We refer to the convex optimization problem which has to be solved at each step in RatioDCA (line 4) as the *inner problem*.

**Proposition 4.1** *The sequence $f^k$ produced by RatioDCA satisfies $F(f^k) > F(f^{k+1})$ for all $k \geq 0$ or the sequence terminates.*

The sequence $F(f^k)$ is not only monotonically decreasing but converges to a generalized nonlinear eigenvector as introduced in [14].

**Theorem 4.1** *Each cluster point $f^*$ of the sequence $f^k$ produced by the RatioDCA is a nonlinear eigenvector with eigenvalue $\lambda^* = \frac{R(f^*)}{S(f^*)} \in \left[0, F(f^0)\right]$ in the sense that it fulfills*

$$0 \in \partial R_1(f^*) - \partial R_2(f^*) - \lambda^*\big(\partial S_1(f^*) - \partial S_2(f^*)\big).$$

*If $S_1 - S_2$ is continuously differentiable at $f^*$, then $F$ has a critical point at $f^*$.*

In the balanced graph cut problem (2) we minimize implicitly over non-constant functions. Thus it is important to guarantee that the RatioDCA for this particular problem always converges to a non-constant vector.

**Lemma 4.1** *For every balanced graph cut problem, the RatioDCA converges to a non-constant $f^*$ given that the initial vector $f^0$ is non-constant.*

Now we are ready to state the following key property of our balanced graph clustering algorithm.

**Theorem 4.2** *Let $(A, \overline{A})$ be a given partition of $V$ and let $S : V \to \mathbb{R}_+$ satisfy one of the conditions stated in Theorem 3.1. If one uses as initialization of RatioDCA, $f^0 = \mathbf{1}_A$, then either RatioDCA terminates after one step or it yields an $f^1$ which after optimal thresholding as in Theorem 3.1 gives a partition $(B, \overline{B})$ which satisfies*

$$\frac{\mathrm{cut}(B, \overline{B})}{\hat{S}(B)} < \frac{\mathrm{cut}(A, \overline{A})}{\hat{S}(A)}.$$

The above "improvement theorem" implies that we can use the result of any other graph partitioning method as initialization. In particular, we can always improve the result of spectral clustering.

### 4.2 Solution of the Convex Inner Optimization Problems

The performance of RatioDCA depends heavily on how fast we can solve the corresponding inner problem. We propose to use a primal-dual algorithm for the inner problem and show experimentally that this approach yields faster convergence than the FISTA method of [5] which was applied in [14]. Let us restrict our attention to the case where $R(f) = R_1(f) = \frac{1}{2} \sum_{i,j=1}^n w_{ij} |f_i - f_j|$ and $S_2 = 0$. In other words, we apply the RatioDCA algorithm to (2) with $S = S_1$ which is what we need, e.g., for the tight relaxations of the Cheeger cut, normalized cut and truncated Cheeger cut families. Hence, the inner problem of the RatioDCA algorithm (line 4) has the form

$$f^{k+1} = \arg\min_{\|u\|_2 \le 1} \{ \frac{1}{2} \sum_{i,j=1}^n w_{ij} |f_i - f_j| - \lambda^k \langle u, s_1(f^k) \rangle \}. \tag{3}$$

Recently, Arrow-Hurwicz-type primal-dual algorithms have become popular, e.g., in image processing, to solve problems whose objective function consists of the sum of convex terms, cf., e.g., [10, 7]. We propose to use the following primal-dual algorithm of [7] where it is referred to as Algorithm 2. We call this method a *primal-dual hybrid gradient algorithm* (PDHG) here since this term is used for similar algorithms in the literature. Note that the operator $P_{\|\cdot\|_\infty \le 1}$ in the first step is the componentwise projection onto the interval $[-1, 1]$. For the sake of readability, we define the linear operator $B : \mathbb{R}^V \to \mathbb{R}^E$ by $Bu = (w_{ij}(u_i - u_j))_{i,j=1}^n$ and its transpose is then $B^{\mathrm{T}} \beta = \left( \sum_{j=1}^n w_{ij} (\beta_{i,j} - \beta_{j,i}) \right)_{i=1}^n$.

---

**Algorithm PDHG** – Solution of the inner problem of RatioDCA for (2) and $S$ convex

1: **Initialization:** $u^0, \bar{u}^0, \beta^0 = 0, \gamma, \sigma_0, \tau_0 > 0$ with $\sigma_0 \tau_0 \le 1/\|B\|_2^2$
2: **repeat**
3:     $\beta^{l+1} = P_{\|\cdot\|_\infty \le 1}(\beta^l + \sigma_l B \bar{u}^l)$
4:     $u^{l+1} = \frac{1}{1+\tau_l} \left( u^l - \tau_l (B^{\mathrm{T}} \beta^{l+1} - 2\lambda^k s_1(f^k)) \right)$
5:     $\theta_l = 1/\sqrt{1 + 2\gamma \tau_l}, \quad \tau_{l+1} = \theta_l \tau_l, \quad \sigma_{l+1} = \sigma_l/\theta_l$
6:     $\bar{u}^{l+1} = u^{l+1} + \theta_l(u^{l+1} - u^l)$
7: **until** duality gap $< \epsilon$
8: **Output:** $f^{k+1} \approx u^{l+1}/\|u^{l+1}\|_2$

---

Although PDHG and FISTA have the same guaranteed converges rates of $\mathcal{O}(1/l^2)$, our experiments show that for clustering applications, PDHG can outperform FISTA substantially. In Fig.1, we illustrate this difference on a toy problem. Note that a single step takes about the same computation time for both algorithms so that the number of iterations is a valid criterion for comparison. In the supplementary material, we also consider the inner problem of RatioDCA for the tight relaxation of the hard balanced cut. Although, in this case we have to deal with $S_2 \ne 0$ in the inner problem of RatioDCA, we can derive a similar PDHG method since the objective function is still the a sum of convex terms.

## 5 Experiments

In a first experiment, we study the influence of the different balancing criteria on the obtained clustering. The data is a Gaussian mixture in $\mathbb{R}^{20}$ where the projection onto the first two dimensions is shown in Figure 2 - the remaining 18 dimensions are just noise. The distribution of the 2000 points is [1200,600,200]. A symmetric $k$-NN-graph with $k = 20$ is built with Gaussian weights $e^{-\frac{2\|x-y\|^2}{\max\{\sigma_{x,k}^2, \sigma_{y,k}^2\}}}$ where $\sigma_{x,k}$ is the $k$-NN distance of point $x$. For better interpretation, we report

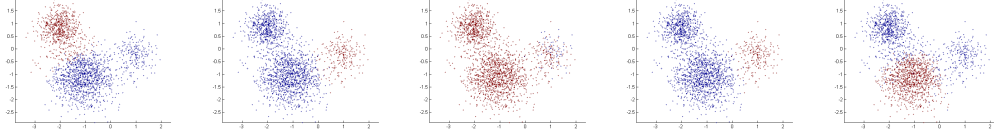

Figure 2: From left to right: Cheeger 1-cut, Normalized 1-cut, truncated Cheeger cut (TCC), hard balanced cut (HBC), hard Cheeger cut (HCC). The criteria are the normalized ones, i.e., the vertex weights are $e_i = d_i$.

all resulting partitions with respect to all balanced graph cut criteria, cut and the size of the largest component in the following table. The parameter for truncated, hard Cheeger cut and hard balanced cut is set to $K = 200$. One observes that the normalized 1-cut results in a less balanced partition but with a much smaller cut than the Cheeger 1-cut, which is itself less balanced than the hard Cheeger cut. The latter is fully balanced but has an even higher cut. The truncated Cheeger cut has a smaller cut than the hard balanced cut but its partition is not feasible. Note that the hard balanced cut is similar to the normalized 1-cut but achieves smaller cut at the prize of a larger maximal component. Thus, the example nicely shows how the different balance criterion influence the final partition.

| Criterion \ Obj. | Cut | $\max\{|A|, |\overline{A}|\}$ | Ch. 1-cut | N. 1-cut | $\text{TCC}_{200}$ | $\text{HBC}_{200}$ | $\text{HCC}_{200}$ |
|---|---|---|---|---|---|---|---|
| Cheeger 1-cut | 408.4 | 1301 | 0.099 | 0.079 | 2.042 | 408.4 | 0.817 |
| Norm. 1-cut | 178.3 | 1775 | 0.132 | 0.075 | 0.892 | 178.3 | 6.858 |
| Trunc. Ch. cut | 153.6 | 1945 | 0.513 | 0.263 | 0.768 | $\infty$ | $\infty$ |
| Hard bal. cut | 175.4 | 1785 | 0.134 | 0.076 | 0.877 | 175.4 | 10.96 |
| Hard Ch. cut | 639.2 | 1000 | 0.119 | 0.115 | 3.196 | 639.2 | 0.798 |

Next we perform unnormalized 1-spectral clustering on the full USPS, normal and extended[1] MNIST-datasets (resp. 9298, 70000 and 630000 points) in the same setting as in [14] with no vertex weights, that is $e_i = 1, \forall i \in V$. As clustering criterion for multi-partitioning we use the multicut version of the normalized 1-cut, given as $\text{RCut}(C_1, \ldots, C_M) = \sum_{i=1}^{M} \frac{\text{cut}(C_i, \overline{C_i})}{|C_i|}$ . We successively subdivide clusters until the desired number of clusters ($M = 10$) is reached. This recursive partitioning scheme is used for all methods. In [14] the Cheeger 1-cut has been used which is not compatible with the multi-cut criterion. We expect that using the normalized 1-cut for the bipartitioning steps we should get better results. The results of the other methods for USPS and MNIST (normal) are taken from [14]. Each bipartitioning step is initialized randomly. Out of 100 obtained multi-partitionings we report the results of the best clustering with respect to the multi-cut criterion. The next table shows the obtained RCut and errors.

| Vertices/Edges | | N. 1-cut | Ch. 1-cut[14] | S.&B.[26] | 1.1-SCl [6] | Standard spectral |
|---|---|---|---|---|---|---|
| USPS | Rcut | 0.6629 | 0.6661 | 0.6663 | 0.6676 | 0.8180 |
| 9K/272K | Error | 0.1301 | 0.1349 | 0.1309 | 0.1308 | 0.1686 |
| MNIST (Normal) | Rcut | 0.1499 | 0.1507 | 0.1545 | 0.1529 | 0.2252 |
| 70K/1043K | Error | 0.1236 | 0.1244 | 0.1318 | 0.1293 | 0.1883 |
| MNIST (Ext) | Rcut | 0.0996 | 0.0997 | – | – | 0.1594 |
| 630K/9192K | Error | 0.1180 | 0.1223 | – | – | 0.2297 |

We see for all datasets improvements in the obtained cut. Also a slight decrease in the obtained error can be observed. The improvements are not so drastic as the clustering is already very good. The problem is that for both datasets one digit is split (0) and two are merged (4 and 9) resulting in seemingly large errors. Similar results hold for the extended MNIST dataset. Note that the resulting error is comparable to recently reported results on semi-supervised learning [18].

## Footnotes

[1]The extended MNIST dataset is generated by translating each original input image of MNIST by one pixel (8 directions).

# References

[1] N. Alon and V. D. Milman. $\lambda_1$, isoperimetric inequalities for graphs, and superconcentrators. *J. Combin. Theory Ser. B*, 38(1):73–88, 1985.

[2] R. Andersen and K. Lang. An algorithm for improving graph partitions. In *Proc. of the 19th ACM-SIAM Symposium on Discrete Algorithms (SODA 2008)*, pages 651–660, 2008.

[3] S. Arora, J. R. Lee, and A. Naor. Expander flows, geometric embeddings and graph partitioning. In *Proc. 36th Annual ACM Symp. on Theory of Computing (STOC)*, pages 222–231. ACM, 2004.

[4] F. Bach. Convex analysis and optimization with submodular functions, 2010. arXiv:1010.4207v2.

[5] A. Beck and M. Teboulle. A fast iterative shrinkage-thresholding algorithm for linear inverse problems. *SIAM J. Imaging Sciences*, 2:183–202, 2009.

[6] T. Bühler and M. Hein. Spectral clustering based on the graph p-Laplacian. In L. Bottou and M. Littman, editors, *Proc. of the 26th Int. Conf. on Machine Learning (ICML)*, pages 81–88. Omnipress, 2009.

[7] A. Chambolle and T. Pock. A first-order primal-dual algorithm for convex problems with applications to imaging. *Journal of Mathematical Imaging and Vision*, 40(1):120–145, 2011.

[8] F. Chung. *Spectral Graph Theory*. AMS, Providence, RI, 1997.

[9] W. E. Donath and A. J. Hoffman. Lower bounds for the partitioning of graphs. *IBM J. Res. Develop.*, 17:420–425, 1973.

[10] E. Esser, X. Zhang, and T. F. Chan. A general framework for a class of first order primal-dual algorithms for convex optimization in imaging science. *SIAM J. Imaging Sciences*, 3(4):1015–1046, 2010.

[11] S. Fujishige. *Submodular functions and optimization*, volume 58 of *Annals of Discrete Mathematics*. Elsevier B. V., Amsterdam, second edition, 2005.

[12] Stephen Guattery and Gary L. Miller. On the quality of spectral separators. *SIAM Journal on Matrix Analysis and Applications*, 19:701–719, 1998.

[13] L. Hagen and A. B. Kahng. Fast spectral methods for ratio cut partitioning and clustering. *Proc. IEEE Intl. Conf. on Computer-Aided Design*, pages 10–13, November 1991.

[14] M. Hein and T. Bühler. An inverse power method for nonlinear eigenproblems with applications in 1-spectral clustering and sparse pca. In *Advances in Neural Information Processing Systems 23 (NIPS 2010)*, pages 847–855, 2010.

[15] J.-B. Hiriart-Urruty. Generalized differentiability, duality and optimization for problems dealing with differences of convex functions. In *Convexity and duality in optimization*, pages 37–70. 1985.

[16] C. Houdré. Mixed and Isoperimetric Estimates on the Log-Sobolev Constants of Graphs and Markov Chains. *Combinatorica*, 21:489–513, 2001.

[17] Y. Kawahara, K. Nagano, and Y. Okamoto. Submodular fractional programming for balanced clustering. *Pattern Recognition Letters*, 32:235–243, 2011.

[18] W. Liu, J. He, and S.-F. Chang. Large graph construction for scalable semi-supervised learning. In *Proc. of the 27th Int. Conf. on Machine Learning (ICML)*, 2010.

[19] L. Lovász. Submodular functions and convexity. In *Mathematical programming: the state of the art (Bonn, 1982)*, pages 235–257. Springer, Berlin, 1983.

[20] M. Maier, U. von Luxburg, and M. Hein. Influence of graph construction on graph-based clustering measures. In *Advances in Neural Information Processing Systems 21 (NIPS)*, pages 1025 – 1032, 2009.

[21] M. Narasimhan and J. Bilmes. A submodular-supermodular procedure with applications to discriminative structure learning. In *21st Conference on Uncertainty in Artificial Intelligence (UAI)*, 2005.

[22] M. Narasimhan and J. Bilmes. Local search for balanced submodular clusterings. In *20th International Joint Conference on Artificial Intelligence (IJCAI)*, 2007.

[23] S. B. Patkar and H. Narayanan. Improving graph partitions using submodular functions. *Discrete Appl. Math.*, 131(2):535–553, 2003.

[24] A. Pothen, H. D. Simon, and K.-P. Liou. Partitioning sparse matrices with eigenvectors of graphs. *SIAM Journal on Matrix Analysis and Applications*, 11:430 – 452, 1990.

[25] O. S. Rothaus. Analytic inequalities, Isoperimetric Inequalities and Logarithmic Sobolev Inequalities. *Journal of Functional Analysis*, 64:296–313, 1985.

[26] A. Szlam and X. Bresson. Total variation and Cheeger cuts. In *Proceedings of the 27th International Conference on Machine Learning*, pages 1039–1046. Omnipress, 2010.

[27] J.-P. Tillich. Edge isoperimetric inequalities for product graphs. *Discrete Mathematics*, 213:291–320, 2000.

[28] U. von Luxburg. A tutorial on spectral clustering. *Statistics and Computing*, 17:395–416, 2007.

